# Active Learning of Model Evidence Using Bayesian Quadrature

**Michael A. Osborne**
University of Oxford
mosb@robots.ox.ac.uk

**David Duvenaud**
University of Cambridge
dkd23@cam.ac.uk

**Roman Garnett**
Carnegie Mellon University
rgarnett@cs.cmu.edu

**Carl E. Rasmussen**
University of Cambridge
cer54@cam.ac.uk

**Stephen J. Roberts**
University of Oxford
sjrob@robots.ox.ac.uk

**Zoubin Ghahramani**
University of Cambridge
zoubin@eng.cam.ac.uk

## Abstract

Numerical integration is a key component of many problems in scientific computing, statistical modelling, and machine learning. Bayesian Quadrature is a model-based method for numerical integration which, relative to standard Monte Carlo methods, offers increased sample efficiency and a more robust estimate of the uncertainty in the estimated integral. We propose a novel Bayesian Quadrature approach for numerical integration when the integrand is non-negative, such as the case of computing the marginal likelihood, predictive distribution, or normalising constant of a probabilistic model. Our approach approximately marginalises the quadrature model's hyperparameters in closed form, and introduces an active learning scheme to optimally select function evaluations, as opposed to using Monte Carlo samples. We demonstrate our method on both a number of synthetic benchmarks and a real scientific problem from astronomy.

## 1 Introduction

The fitting of complex models to big data often requires computationally intractable integrals to be approximated. In particular, machine learning applications often require integrals over probabilities

$$Z = \langle \ell \rangle = \int \ell(\mathbf{x}) p(\mathbf{x}) \mathrm{d}\mathbf{x}, \tag{1}$$

where $\ell(\mathbf{x})$ is non-negative. Examples include computing marginal likelihoods, partition functions, predictive distributions at test points, and integrating over (latent) variables or parameters in a model. While the methods we will describe are applicable to all such problems, we will explicitly consider computing model evidences, where $\ell(\mathbf{x})$ is the unnormalised likelihood of some parameters $x_1, \ldots, x_D$. This is a particular challenge in modelling big data, where evaluating the likelihood over the entire dataset is extremely computationally demanding.

There exist several standard randomised methods for computing model evidence, such as annealed importance sampling (AIS) [1], nested sampling [2] and bridge sampling. For a review, see [3]. These methods estimate $Z$ given the value of the integrand on a set of sample points, whose size is limited by the expense of evaluating $\ell(\mathbf{x})$. It is well known that convergence diagnostics are often unreliable for Monte Carlo estimates of partition functions [4, 5, 6]. Most such algorithms also have parameters which must be set by hand, such as proposal distributions or annealing schedules.

An alternative, model-based, approach is Bayesian Quadrature (BQ) [7, 8, 9, 10], which specifies a distribution over likelihood functions, using observations of the likelihood to infer a distribution

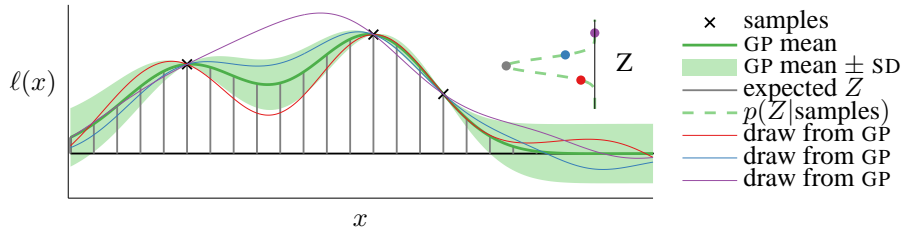

Figure 1: Model-based integration computes a posterior for the integral $Z = \int \ell(\mathbf{x})p(\mathbf{x})\mathrm{d}\mathbf{x}$, conditioned on sampled values of the function $\ell(\mathbf{x})$. For this plot, we assume a Gaussian process model for $\ell(\mathbf{x})$ and a broad Gaussian prior $p(x)$. The variously probable integrands permitted under the model will give different possible values for $Z$, with associated differing probabilities.

for $Z$ (see Figure 1). This approach offers improved sample efficiency [10], crucial for expensive samples computed on big data. We improve upon this existing work in three ways:

**Log-GP:** [10] used a GP prior on the likelihood function; this is a poor model in this case, unable to express the non-negativity and high dynamic range of most likelihood functions. [11] introduced an approximate means of exploiting a GP on the logarithm of a function (henceforth, a log-GP), which better captures these properties of likelihood functions. We apply this method to estimate $Z$, and extend it to compute $Z$'s posterior variance and expected variance after adding a sample.

**Active Sampling:** Previous work on BQ has used randomised or *a priori* fixed sampling schedules. We use active sampling, selecting locations which minimise the expected uncertainty in $Z$.

**Hyperparameter Marginalisation:** Uncertainty in the hyperparameters of the model used for quadrature has previously been ignored, leading to overconfidence in the estimate of $Z$. We introduce a tractable approximate marginalisation of input scale hyperparameters.

From a Bayesian perspective, numerical integration is fundamentally an inference and sequential decision making problem: Given a set of function evaluations, what can we infer about the integral, and how do we decide where to next evaluate the function. Monte Carlo methods, including MCMC, provide simple but generally suboptimal and non-adaptive answers: compute a sample mean, and evaluate randomly. Our approach attempts to learn about the integrand as it evaluates the function at different points, and decide based on information gain where to evaluate next. We compare our approach against standard Monte Carlo techniques and previous Bayesian approaches on both simulated and real problems.

## 2   Bayesian Quadrature

*Bayesian quadrature* [8, 10] is a means of performing Bayesian inference about the value of a potentially nonanalytic integral, $\langle f \rangle := \int f(x)p(x)\mathrm{d}x$. For clarity, we henceforth assume the domain of integration $\mathcal{X} = \mathbb{R}$, although all results generalise to $\mathbb{R}^n$. We assume a Gaussian density $p(x) := \mathcal{N}(x; \nu_x, \lambda_x)$, although other convenient forms, or, if necessary, the use of an importance re-weighting trick ($q(x) = {q(x)}/{p(x)}p(x)$ for any $q(x)$), allow any other integral to be approximated.

Quadrature involves evaluating $f(x)$ at a vector of sample points $\mathbf{x}_s$, giving $\boldsymbol{f}_s := f(\mathbf{x}_s)$. Often this evaluation is computationally expensive; the consequent sparsity of samples introduces uncertainty about the function $f$ between them, and hence uncertainty about the integral $\langle f \rangle$.

Previous work on BQ chooses a Gaussian process (GP) [12] prior for $f$, with mean $\mu_f$ and Gaussian covariance function

$$K(x_1, x_2) := h^2 \mathcal{N}(x_1; x_2, w). \tag{2}$$

Here hyperparameter $h$ species the output scale, while hyperparameter $w$ defines a (squared) input scale over $x$. These scales are typically fitted using type two maximum likelihood (MLII); we will later introduce an approximate means of marginalising them in Section 4. We'll use the following dense notation for the standard GP expressions for the posterior mean $m$, covariance $C$, and variance

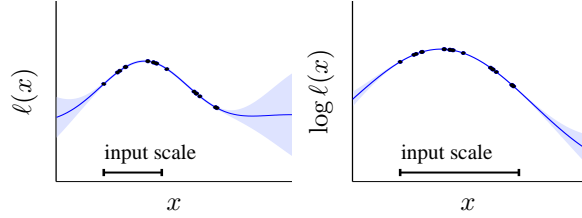

Figure 2: A GP fitted to a peaked log-likelihood function is typically a better model than GP fit to the likelihood function (which is non-negative and has high dynamic range). The former GP also usually has the longer input scale, allowing it to generalise better to distant parts of the function.

$V$, respectively: $m_{f|s}(x_\star) := m(f_\star|\boldsymbol{f}_s)$, $C_{f|s}(x_\star, x'_\star) := C(f_\star, f'_\star|\boldsymbol{f}_s)$ and $V_{f|s}(x_\star) := V(f_\star|\boldsymbol{f}_s)$. Note that this notation assumes implicit conditioning on hyperparameters. Where required for disambiguation, we'll make this explicit, as per $m_{f|s,w}(x_\star) := m(f_\star|\boldsymbol{f}_s, w)$ and so forth.

Variables possessing a multivariate Gaussian distribution are jointly Gaussian distributed with any affine transformations of those variables. Because integration is affine, we can hence use computed samples $\boldsymbol{f}_s$ to perform analytic Gaussian process inference about the value of integrals over $f(x)$, such as $\langle f \rangle$. The mean estimate for $\langle f \rangle$ given $\boldsymbol{f}_s$ is

$$
\begin{aligned}
m(\langle f \rangle|\boldsymbol{f}_s) &= \iint \langle f \rangle\, p(\langle f \rangle|f)\, p(f|\boldsymbol{f}_s)\, \mathrm{d}\langle f \rangle\, \mathrm{d}f \\
&= \iint \langle f \rangle\, \delta\left(\langle f \rangle - \int f(x)\, p(x)\, \mathrm{d}x\right) \mathcal{N}\big(f; m_{f|s}, C_{f|s}\big)\, \mathrm{d}\langle f \rangle\, \mathrm{d}f \\
&= \int m_{f|s}(x)\, p(x)\, \mathrm{d}x,
\end{aligned}
\tag{3}
$$

which is expressible in closed-form due to standard Gaussian identities [10]. The corresponding closed-form expression for the posterior variance of $\langle f \rangle$ lends itself as a natural convergence diagnostic. Similarly, we can compute the posteriors for integrals over the product of multiple, independent functions. For example, we can calculate the posterior mean $m(\langle fg \rangle|\boldsymbol{f}_s, \boldsymbol{g}_s)$ for an integral $\int f(x)g(x)p(x)\mathrm{d}x$. In the following three sections, we will expand upon the improvements this paper introduces in the use of Bayesian Quadrature for computing model evidences.

## 3 Modelling Likelihood Functions

We wish to evaluate the evidence (1), an integral over non-negative likelihoods, $\ell(x)$. Assigning a standard GP prior to $\ell(x)$ ignores prior information about the range and non-negativity of $\ell(x)$, leading to pathologies such as potentially negative evidences (as observed in [10]). A much better prior would be a GP prior on $\log \ell(x)$ (see Figure 2). However, the resulting integral is intractable,

$$
m(Z|\log \boldsymbol{\ell}_s) = \int \left(\int \exp\big(\log \ell(x)\big)p(x)\, \mathrm{d}x\right) \mathcal{N}\big(\log \ell; m_{\log \ell|s}, C_{\log \ell|s}\big)\, \mathrm{d}\log \ell,
\tag{4}
$$

as (4) does not possess the affine property exploited in (3). To progress, we adopt an approximate inference method inspired by [11] to tractably integrate under a log-GP prior.[1] Specifically, we linearise the problematic exponential term around some point $\log \ell_0(x)$, as

$$
\exp\big(\log \ell(x)\big) \simeq \exp\big(\log \ell_0(x)\big) + \exp\big(\log \ell_0(x)\big)\big(\log \ell(x) - \log \ell_0(x)\big)
\tag{5}
$$

The integral (4) consists of the product of $Z$ and a GP for $\log \ell$. The former is $\sim \exp \log \ell$, the latter is $\sim \exp\big(-(\log \ell - m)^2\big)$, effectively permitting only a small range of $\log \ell$ functions. Over this narrow region, it is reasonable to assume that $Z$ does not vary too dramatically, and can be approximated as linear in $\log \ell$, as is assumed by (5). Using this approximation, and making the definition $\Delta_{\log \ell|s} := m_{\log \ell|s} - \log \ell_0$, we arrive at

$$
m(Z|\log \boldsymbol{\ell}_s) \simeq m(Z|\log \ell_0, \log \boldsymbol{\ell}_s) := \int \ell_0(x)p(x)\, \mathrm{d}x + \int \ell_0(x)\Delta_{\log \ell|s}(x)p(x)\, \mathrm{d}x.
\tag{6}
$$

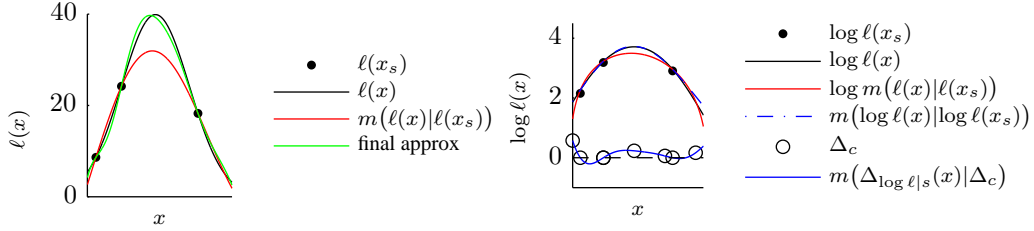

Figure 3: Our approximate use of a GP for $\log \ell(x)$ improves upon the use of a GP for $\ell(x)$ alone. Here the 'final approx' is $m_{\ell|s}(1 + \Delta_{\log \ell|s})$, from (5) and (6).

We now choose $\ell_0$ to allow us to resolve the first integral in (6). First, we introduce a secondary GP model for $\ell$, the non-log space, and choose $\ell_0 := m_{\ell|s}$, where $m_{\ell|s}$ is the standard GP conditional mean for $\ell$ given observations $\ell(\mathbf{x}_s)$. For both GPs[2] (over both log and non-log spaces), we take zero prior means and Gaussian covariances of the form (2). It is reasonable to use zero prior means: $\ell(x)$ is expected to be negligible except at a small number of peaks. If a quantity is dependent upon the GP prior for $\ell$, it will be represented as conditional on $\boldsymbol{\ell}_s$; if dependent upon the former GP prior over $\log \ell$, it will be conditional upon $\log \boldsymbol{\ell}_s$. We expect $\Delta_{\log \ell|s}(x)$ to be small everywhere relative to the magnitude of $\log \ell(x)$ (see Figure 3). Hence $\log \ell_0$ is close to the peaks of the Gaussian over $\log \ell$, rendering our linearisation appropriate. For $\ell_0$, the first integral in (6) becomes tractable.

Unfortunately, the second integral in (6) is non-analytic due to the $\log \ell_0$ term within $\Delta_{\log \ell|s}$. As such, we perform another stage of Bayesian quadrature by treating $\Delta_{\log \ell|s}$ as an unknown function of $x$. For tractability, we assume this prior is independent of the prior for $\log \ell$. We use another GP for $\Delta_{\log \ell|s}$, with zero prior mean and Gaussian covariance (2). A zero prior mean here is reasonable: $\Delta_{\log \ell|s}$ is exactly zero at $\mathbf{x}_s$, and tends to zero far away from $\mathbf{x}_s$, where both $m_{\log \ell|s}$ and $\log \ell_0$ are given by the compatible prior means for $\log \ell$ and $\ell$. We must now choose *candidate points* $\mathbf{x}_c$ at which to evaluate the $\Delta_{\log \ell|s}$ function (note we do not need to evaluate $\ell(x_c)$ in order to compute $\Delta_c := \Delta_{\log \ell|s}(x_c)$). $\mathbf{x}_c$ should firstly include $\mathbf{x}_s$, where we know that $\Delta_{\log \ell|s}$ is equal to zero. We select the remainder of $\mathbf{x}_c$ at random on the hyper-ellipses (whose axes are defined by the input scales for $\ell$) surrounding existing observations; we expect $\Delta_{\log \ell|s}$ to be extremised at such $\mathbf{x}_c$. We limit ourselves to a number of candidates that scales linearly with the dimensionality of the integral for all experiments.

Given these candidates, we can now marginalise (6) over $\Delta_{\log \ell|s}$ to give

$$m(Z|\log \boldsymbol{\ell}_s) \simeq m(Z|\log \ell_0, \log \boldsymbol{\ell}_s, \Delta_c) = m(Z|\boldsymbol{\ell}_s) + m\big(\langle \ell \Delta_{\log \ell|s} \rangle \big| \boldsymbol{\ell}_s, \Delta_c\big), \qquad (7)$$

where both terms are analytic as per Section 2; $m(Z|\boldsymbol{\ell}_s)$ is of the form (3). The correction factor, the second term in (7), is expected to be small, since $\Delta_{\log \ell|s}$ is small. We extend the work of [11] to additionally calculate the variance in the evidence,

$$V(Z|\log \ell_0, \log \boldsymbol{\ell}_s, \Delta_c) = S(Z\,|\log \ell_0, \log \boldsymbol{\ell}_s) - m(Z|\log \ell_0, \log \boldsymbol{\ell}_s, \Delta_c)^2 , \qquad (8)$$

where the second moment is

$$S(Z\,|\log \ell_0, \log \boldsymbol{\ell}_s) := m\big(\langle \ell C_{\log \ell|s} \ell \rangle \big|\log \boldsymbol{\ell}_s\big) + m(Z|\log \ell_0, \log \boldsymbol{\ell}_s, \Delta_c)^2 , \qquad (9)$$

and hence

$$V(Z|\log \ell_0, \log \boldsymbol{\ell}_s, \Delta_c) = m\big(\langle \ell C_{\log \ell|s} \ell \rangle \big|\log \boldsymbol{\ell}_s\big)$$
$$:= \iint m_{\ell|s}(x) m_{\ell|s}(x') C_{\log \ell|s}(x, x') p(x) p(x') \mathrm{d}x \mathrm{d}x', \qquad (10)$$

which is expressible in closed form, although space precludes us from doing so. This variance can be employed as a convergence diagnostic; it describes our uncertainty in the model evidence $Z$.

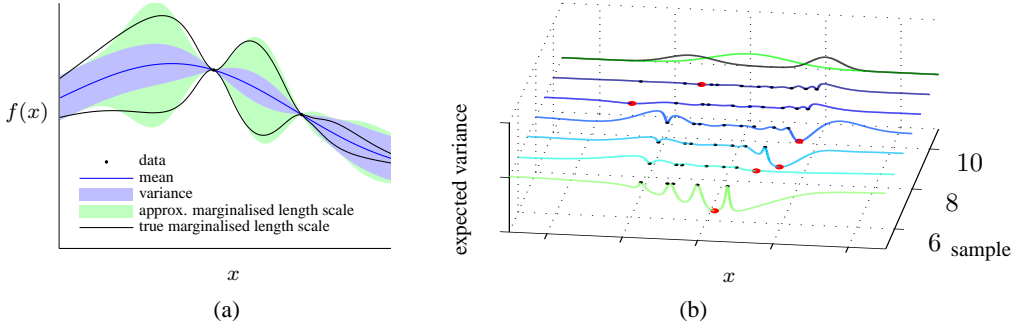

(a)                                           (b)

Figure 4: a) Integrating hyperparameters increases the marginal posterior variance (in regions whose mean varies as the input scales change) to more closely match the true posterior marginal variance. b) An example showing the expected uncertainty in the evidence after observing the likelihood function at that location. $p(x)$ and $l(x)$ are plotted at the top in green and black respectively, the next sample location in red. Note the model discovering a new mode on the right hand side, sampling around it, then moving on to other regions of high uncertainty on the left hand side.

In summary, we have described a linearisation approach to exploiting a GP prior over log-likelihoods; this permitted the calculation of the analytic posterior mean (7) and variance (10) of $Z$. Note that our approximation will improve with increasing numbers of samples: $\Delta_{\log \ell|s}$ will eventually be small everywhere, since it is clamped to zero at each observation. The quality of the linearisation can also be improved by increasing the number of candidate locations, at the cost of slower computation.

## 4  Marginalising hyperparameters

We now present a novel means of approximately marginalising the hyperparameters of the GP used to model the log-integrand, $\log \ell$. In previous approaches to Bayesian Quadrature, hyperparameters were estimated using MLII, which approximates the likelihood as a delta function. However, ignoring the uncertainty in the hyperparameters can lead to pathologies. In particular, the reliability of the variance for $Z$ depends crucially upon marginalising over all unknown quantities.

The hyperparameters of most interest are the input scales $w$ for the GP over the log-likelihood; these hyperparameters can have a powerful influence on the fit to a function. We use MLII to fit all hyperparameters other than $w$. Marginalisation of $w$ is confounded by the complex dependence of our predictions upon these input scales. We make the following essential approximations:

**Flat prior:** We assume that the prior for $w$ is broad, so that our posterior is the normalised likelihood.

**Laplace approximation:** $p(\log \ell_s|w)$ is taken as Gaussian with mean equal to the MLII value $\hat{w}$ and with diagonal covariance $C_w$, diagonal elements fitted using the second derivatives of the likelihood. We represent the posterior mean for $\log \ell$ conditioned on $\hat{w}$ as $\hat{m} := m_{\log \ell|s,\hat{w}}$.

**GP mean affine in $w$:** Given the narrow width of the likelihood for $w$, $p(\log \ell|\log \ell_s, w)$ is approximated as having a GP mean which is affine in $w$ around the MLII values, and a constant covariance; $m_{\log \ell|s,w} \simeq \hat{m} + \frac{\partial \hat{m}}{\partial w}(w - \hat{w})$ and $C_{\log \ell|s,w} \simeq C_{\log \ell|s,\hat{w}}$.

The implication of these approximations is that the marginal posterior mean over $\log \ell$ is simply $\tilde{m}_{\log \ell|s} := m_{\log \ell|s,\hat{w}}$. The marginal posterior variance is $\tilde{C}_{\log \ell|s} := C_{\log \ell|s,\hat{w}} + \frac{\partial \hat{m}}{\partial w} C_w \frac{\partial \hat{m}}{\partial w}$. An example of our approximate posterior is depicted in Figure 4a. Our approximations give the marginal posterior mean for $Z$:

$$\tilde{m}(Z|\log \ell_0, \log \boldsymbol{\ell}_s, \Delta_c) := m(Z|\log \ell_0, \log \boldsymbol{\ell}_s, \Delta_c, \hat{w}) , \tag{11}$$

of the form (7). The marginal posterior variance

$$\tilde{V}(Z|\log \ell_0, \log \boldsymbol{\ell}_s, \Delta_c) = \iint \mathrm{d}x\, \mathrm{d}x'\, m_{\ell|s}(x)\, m_{\ell|s}(x') \left( C_{\log \ell|s}(x, x') + \frac{\partial \hat{m}(x)}{\partial w}\, C_w\, \frac{\partial \hat{m}(x')}{\partial w} \right) \tag{12}$$

is possible, although laborious, to express analytically, as with (10).

# 5 Active Sampling

One major benefit of model-based integration is that samples can be chosen by any method, in contrast to Monte Carlo methods, which typically must sample from a specific distribution. In this section, we describe a scheme to select samples $x_s$ sequentially, by minimising the *expected* uncertainty in the evidence that remains after taking each additional sample.[3] We take the variance in the evidence as our loss function, and proceed according to Bayesian decision theory.

Surprisingly, the posterior variance of a GP model with fixed hyperparameters does not depend on the function values at sampled locations at all; only the location of those samples matters. In traditional Bayesian quadrature, the evidence is an affine transformation of the sampled likelihood values, hence its estimate for the variance in the evidence is also independent of likelihood values. As such, active learning with fixed hyperparameters is pointless, and the optimal sampling design can be found in advance [13].

In Section 3, we took $Z$ as an affine transform of the log-likelihood, which we model with a GP. As the affine transformation (5) itself depends on the function values (via the dependence of $\log \ell_0$), the conclusions of the previous paragraph do not apply, and active learning is desirable. The uncertainty over the hyperparameters of the GP further motivates active learning: without assuming *a priori* knowledge of the hyperparameters, we can't evaluate the GP to precompute a sampling schedule. The approximate marginalisation of hyperparameters permits an approach to active sampling that acknowledges the influence new samples may have on the posterior over hyperparameters.

Active sampling selects a new sample $x_a$ so as to minimise the expected variance in the evidence after adding the sample to the model of $\ell$. The objective is therefore to choose the $x_a$ that minimises the expected loss; $x_a = \mathrm{argmin}_{x_a} \langle V(Z | \log \ell_0, \log \boldsymbol{\ell}_{s,a}) \mid \log \ell_0, \log \boldsymbol{\ell}_s \rangle$ (note $x_a$ is implicitly conditioned on, as usual for function inputs) where the expected loss is

$$\langle V(Z | \log \ell_0, \log \boldsymbol{\ell}_{s,a}) \mid \log \ell_0, \log \boldsymbol{\ell}_s \rangle = S(Z | \log \ell_0, \log \boldsymbol{\ell}_s) - \int m(Z | \log \ell_0, \log \boldsymbol{\ell}_{a,s}, \Delta_c)^2$$
$$\times \mathcal{N}\left( \log \ell_a; \hat{m}_a, \hat{C}_a + \frac{\partial \hat{m}_a}{\partial w} C_w \frac{\partial \hat{m}_a^\mathsf{T}}{\partial w} \right) \mathrm{dlog} \ell_a, \quad (13)$$

and we define $\hat{m}_a := m(\log \ell_a | \log \boldsymbol{\ell}_s, \hat{w})$ and $\hat{C}_a := V(\log \ell_a | \log \boldsymbol{\ell}_s, \hat{w})$. The first term in (13), the second moment, is independent of the selection of $x_a$ and can hence be safely ignored for active sampling (true regardless of the model chosen for the likelihood). The second term, the negative expected squared mean, can be resolved analytically[4] for any trial $x_a$ (we omit the laborious details).

Importantly, we do not have to make a linearisation approximation for this new sample. That is, the GP posterior over $\log \ell_a$ can be fully exploited when performing active sampling.

In order to minimise the expected variance, the objective in (13) encourages the maximisation of the expected squared mean of $Z$. Due to our log-GP model, one means the method can use to do this is to seek points where the log-likelihood is predicted to be large: which we call *exploitation*. The objective in (13) naturally balances exploitation against *exploration*: the choice of points where our current variance in the log-likelihood is significant (see Figure 4b). Note that the variance for $\log \ell_a$ is increased by approximate integration over hyperparameters, encouraging exploration.

# 6 Experiments

We now present empirical evaluation of our algorithm in a variety of different experiments.

**Metrics:** We judged our methods according to three metrics, all averages over $N$ similar experiments indexed by $i$. Define $Z_i$ as the ground truth evidence for the $i$th experiment, $m(Z_i)$ as its estimated mean and $V(Z_i)$ as its predicted variance. Firstly, we computed the average log error,

ALE $:= \frac{1}{N} \sum_{i=1}^{N} |\log m(Z_i) - \log Z_i|$ . Next we computed the negative log-density of the truth, assuming experiments are independent, $-\log p(\boldsymbol{Z}) = -\sum_{i=1}^{N} \log \mathcal{N}(Z_i; m(Z_i), V(Z_i))$, which quantifies the accuracy of our variance estimates. We also computed the calibration $\mathcal{C}$, defined as the fraction of experiments in which the ground truth lay within our 50% confidence interval $\left(m(Z_i) - 0.6745\sqrt{V(Z_i)}, m(Z_i) + 0.6745\sqrt{V(Z_i)}\right)$. Ideally, $\mathcal{C}$ would be 50%: any higher, and a method is under-confident, any lower and it is over-confident.

**Methods:** We first compared against simple Monte Carlo (SMC). SMC generates samples $x_1, \dots, x_N$ from $p(x)$, and estimates $Z$ by $\hat{Z} = 1/N \sum_{n=1}^{N} \ell(x_n)$. An estimate of the variance of $\hat{Z}$ is given by the standard error of $\ell(\mathbf{x})$. As an alternative Monte Carlo technique, we implemented Annealed Importance Sampling (AIS) using a Metropolis-Hastings sampler. The inverse temperature schedule was linear as in [10], and the proposal width was adjusted to attain approximately a 50% acceptance rate. Note that a single AIS chain provides no ready means of determining the posterior variance for its estimate of $Z$. Our first model-based method was Bayesian Monte Carlo (BMC) – the algorithm used in [10]. Here samples were drawn from the AIS chain above, and a GP was fit to the likelihood samples. For this and other methods, where not otherwise mentioned, GP hyperparameters were selected using MLII.

We then tested four novel methods. Firstly, Bayesian Quadrature (BQ), which employed the linearisation approach of Section 3 to modeling the log-transformed likelihood values. The samples supplied to it were drawn from the same AIS chain as used above, and 400 candidate points were permitted. BQ* is the same algorithm as BQ but with hyperparameters approximately marginalised, as per Section 4. Note that this influences only the variance of the estimate; the means for BQ and BQ* are identical. The performance of these methods allow us to quantify to what extent our innovations improve estimation given a fixed set of samples.

Next, we tested a novel algorithm, Doubly Bayesian Quadrature (BBQ). The method is so named for the fact that we use not only Bayesian inference (with a GP over the log-transformed likelihood) to compute the posterior for the evidence, but also Bayesian decision theory to select our samples actively, as described in Section 5. BBQ* is identical, but with hyperparameters approximately marginalised. Both algorithms demonstrate the influence of active sampling on our performance.

**Problems:** We used these methods to evaluate evidences given Gaussian priors and a variety of likelihood functions. As in [10] and [11], we focus on low numbers of samples; we permitted tested methods 150 samples on synthetic integrands, and 300 when using real data. We are motivated by real-world, big-data, problems where evaluating likelihood samples is expensive, making it desirable to determine the techniques for evidence estimation that can operate best when permitted only a small number of samples. Ground truth $Z$ is available for some integrals; for the non-analytic integrals, $Z$ was estimated by a run of SMC with $10^5$ samples.

We considered seven synthetic examples. We firstly tested using single Gaussians, in one, four, ten and twenty dimensions. We also tested on mixtures of two Gaussians in one dimension (two examples, alternately widely separated and overlapping) and four dimensions (a single example).

We additionally tested methods on a real scientific problem: detecting a damped Lyman-$\alpha$ absorber (DLA) between the Earth and an observed quasar from spectrographic readings of the quasar. DLAs are large objects consisting primarily of neutral hydrogen gas. The statistical properties of DLAs inform us about the distribution of neutral hydrogen in the universe, which is of fundamental cosmological importance. We model the quasar spectra using a GP; the presence of a DLA is represented as an observation fault with known dynamics [14]. This model has five hyperparameters to be marginalised, to which we assign priors drawn from the large corpus of data obtained from the Sloan Digital Sky Survey (SDSS) [15]. We tested over four datasets; the expense of evaluating a GP likelihood sample on the large datasets available from the SDSS (140TB of data have been released in total) motivates the small sample sizes considered.

**Evaluation** Table 1 shows combined performance on the synthetic integrands listed above. The calibration scores $\mathcal{C}$ show that all methods[5] are systematically overconfident, although our approaches are at least as well calibrated as alternatives. On average, BBQ* provides an estimate

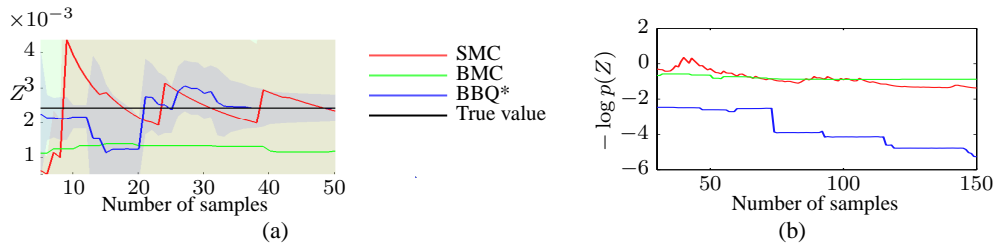

Figure 5: a) The posterior distribution over $Z$ for several methods on a one-dimensional example as the number of samples increases. Shaded regions denote $\pm 2$ SD's from the mean. The shaded regions for SMC and BMC are off the vertical scale of this figure. b) The log density of the true evidence for different methods (colours identical to those in a), compared to the true $Z$ (in black). The integrand is the same as that in Figure 4b.

Table 1: Combined Synthetic Results

| Method | $-\log p(\mathbf{Z})$ | ALE | $\mathcal{C}$ |
|---|---|---|---|
| SMC | $> 1000$ | 1.101 | 0.286 |
| AIS | N/A | 1.695 | N/A |
| BMC | $> 1000$ | 2.695 | 0.143 |
| BQ | $> 1000$ | 6.760 | 0.429 |
| BQ* | $> 1000$ | 6.760 | 0.429 |
| BBQ | 13.597 | 0.919 | 0.286 |
| BBQ* | $-\mathbf{11.909}$ | **0.271** | 0.286 |

Table 2: Combined Real Results

| Method | $-\log p(\mathbf{Z})$ | ALE | $\mathcal{C}$ |
|---|---|---|---|
| SMC | 5.001 | 0.632 | 0.250 |
| AIS | N/A | 2.146 | N/A |
| BMC | 9.536 | 1.455 | 0.500 |
| BQ | 37.017 | 0.635 | 0.000 |
| BQ* | 33.040 | 0.635 | 0.000 |
| BBQ | **3.734** | **0.400** | 0.000 |
| BBQ* | 74.242 | 1.732 | 0.250 |

of $Z$ which is closer to the truth than the other methods given the same number of samples, and assigns much higher likelihood to the true value of $Z$. BBQ* also achieved the lowest error on five, and best likelihood on six, of the seven problems, including the twenty dimensional problem for both metrics. Figure 5a shows a case where both SMC and BBQ* are relatively close to the true value, however BBQ*'s posterior variance is much smaller. Figure 5b demonstrates the typical behaviour of the active sampling of BBQ*, which quickly concentrates the posterior distribution at the true $Z$. The negative likelihoods of BQ* are for every problem slightly lower than for BQ ($-\log p(\mathbf{Z})$ is on average 0.2 lower), indicating that the approximate marginalisation of hyperparameters grants a small improvement in variance estimate.

Table 2 shows results for the various methods on the real integration problems. Here BBQ is clearly the best performer; the additional exploration induced by the hyperparameter marginalisation of BBQ* may have led to local peaks being incompletely exploited. Exploration in a relatively high dimensional, multi-modal space is inherently risky; nonetheless, BBQ* achieved lower error than BBQ on two of the problems.

# 7 Conclusions

In this paper, we have made several advances to the BQ method for evidence estimation. These are: approximately imposing a positivity constraint[6], approximately marginalising hyperparameters, and using active sampling to select the location of function evaluations. Of these contributions, the active learning approach yielded the most significant gains for integral estimation.

**Acknowledgements**

M.A.O. was funded by the ORCHID project (http://www.orchid.ac.uk/).

## Footnotes

[1]In practice, we use the transform $\log\big(\ell(x) + 1\big)$, allowing us to assume the transformed quantity has zero mean. For the sake of simplicity, we omit this detail in the following derivations.

[2]Note that separately modelling $\ell$ and $\log \ell$ is not inconsistent: we use the posterior mean of the GP for $\ell$ only as a convenient parameterisation for $\ell_0$; we do not treat this GP as a full probabilistic model. While this modelling choice may seem excessive, this approach provides significant advantages in the sampling efficiency of the overall algorithm by approximately capturing the non-negativity of our integrand and allowing active sampling.

[3]We also expect such samples to be useful not just for estimating the evidence, but also for any other related expectations, such as would be required to perform prediction using the model.

[4]Here we use the fact that $\int \exp(c\,y) \mathcal{N}(y; m, \sigma^2) \,\mathrm{d}y = \exp(c\,m + 1/2\,c^2\sigma^2)$. We assume that $\Delta_{\log \ell | s}$ does not depend on $\log \ell_a$, only its location $x_a$: we know $\Delta(x_a) = 0$ and assume $\Delta_{\log \ell | s}$ elsewhere remains unchanged.

[5]Because a single AIS chain gives no estimate of uncertainty, it has no likelihood or calibration scores.

[6]Our approximations mean that we cannot guarantee non-negativity, but our approach improves upon alternatives that make no attempt to enforce the non-negativity constraint.

# References

[1] R.M. Neal. Annealed importance sampling. *Statistics and Computing*, 11(2):125–139, 2001.

[2] J. Skilling. Nested sampling. *Bayesian inference and maximum entropy methods in science and engineering*, 735:395–405, 2004.

[3] M.H. Chen, Q.M. Shao, and J.G. Ibrahim. *Monte Carlo methods in Bayesian computation*. Springer, 2000.

[4] R. M. Neal. Probabilistic inference using Markov chain Monte Carlo methods. Technical Report CRG-TR-93-1, University of Toronto, 1993.

[5] S.P. Brooks and G.O. Roberts. Convergence assessment techniques for Markov chain Monte Carlo. *Statistics and Computing*, 8(4):319–335, 1998.

[6] M.K. Cowles, G.O. Roberts, and J.S. Rosenthal. Possible biases induced by MCMC convergence diagnostics. *Journal of Statistical Computation and Simulation*, 64(1):87, 1999.

[7] P. Diaconis. Bayesian numerical analysis. In S. Gupta J. Berger, editor, *Statistical Decision Theory and Related Topics IV*, volume 1, pages 163–175. Springer-Verlag, New York, 1988.

[8] A. O'Hagan. Bayes-Hermite quadrature. *Journal of Statistical Planning and Inference*, 29:245–260, 1991.

[9] M. Kennedy. Bayesian quadrature with non-normal approximating functions. *Statistics and Computing*, 8(4):365–375, 1998.

[10] C. E. Rasmussen and Z. Ghahramani. Bayesian Monte Carlo. In S. Becker and K. Obermayer, editors, *Advances in Neural Information Processing Systems*, volume 15. MIT Press, Cambridge, MA, 2003.

[11] M.A. Osborne, R. Garnett, S.J. Roberts, C. Hart, S. Aigrain, N.P. Gibson, and S. Aigrain. Bayesian quadrature for ratios. In *Proceedings of the Fifteenth International Conference on Artificial Intelligence and Statistics (AISTATS 2012)*, 2012.

[12] C. E. Rasmussen and C. K. I. Williams. *Gaussian Processes for Machine Learning*. MIT Press, 2006.

[13] T. P. Minka. Deriving quadrature rules from Gaussian processes. Technical report, Statistics Department, Carnegie Mellon University, 2000.

[14] R. Garnett, M.A. Osborne, S. Reece, A. Rogers, and S.J. Roberts. Sequential bayesian prediction in the presence of changepoints and faults. *The Computer Journal*, 53(9):1430, 2010.

[15] Sloan Digital Sky Survey, 2011. `http://www.sdss.org/`.

